# Hierarchical Mixtures of Experts Methodology Applied to Continuous Speech Recognition

Ying Zhao, Richard Schwartz, Jason Sroka**, John Makhoul
BBN System and Technologies
70 Fawcett Street
Cambridge MA 02138

## Abstract

In this paper, we incorporate the Hierarchical Mixtures of Experts (HME) method of probability estimation, developed by Jordan [1], into an HMM-based continuous speech recognition system. The resulting system can be thought of as a continuous-density HMM system, but instead of using gaussian mixtures, the HME system employs a large set of hierarchically organized but relatively small neural networks to perform the probability density estimation. The hierarchical structure is reminiscent of a decision tree except for two important differences: each "expert" or neural net performs a "soft" decision rather than a hard decision, and, unlike ordinary decision trees, the parameters of all the neural nets in the HME are automatically trainable using the EM algorithm. We report results on the ARPA 5,000-word and 40,000-word Wall Street Journal corpus using HME models.

## 1  Introduction

Recent research has shown that a continuous-density HMM (CD-HMM) system can out-perform a more constrained tied-mixture HMM system for large-vocabulary continuous speech recognition (CSR) when a large amount of training data is available [2]. In other work, the utility of decision trees has been demonstrated in classification problems by using the "divide and conquer" paradigm effectively, where a problem is divided into a hierarchical set of simpler problems. We present here a new CD-HMM system which

**MIT, Cambridge MA 02139

has similar properties and possesses the same advantages as decision trees, but has the additional important advantage of having automatically trainable "soft" decision boundaries.

## 2   Hierarchical Mixtures of Experts

The method of Hierarchical Mixtures of Experts (HME) developed recently by Jordan [1] breaks a large scale task into many small ones by partitioning the input space into a nested set of regions, then building a simple but specific model (local expert) in each region. The idea behind this method follows the principle of divide-and-conquer which has been utilized in certain approaches to classification problems, such as decision trees. In the decision tree approach, at each level of the tree, the data are divided explicitly into regions. In contrast, the HME model makes use of "soft" splits of the data, i.e., instead of the data being explicitly divided into regions, the data may lie simultaneously in multiple regions with certain probabilities. Therefore, the variance-increasing effect of lopping off distant data in the decision tree can be ameliorated. Furthermore, the "hard" boundaries in the decision tree are fixed once a decision is made, while the "soft" boundaries in the HME are parameterized with generalized sigmoidal functions, which can be adjusted automatically using the Expectation-Maximization (EM) algorithm during the splitting.

Now we describe how to apply the HME methodology to the CSR problem. For each state of a phonetic HMM, a separate HME is used to estimate the likelihood. The actual HME first computes a posterior probability $P(l|x, s)$, the probability of phoneme class $l$, given the input feature vector $x$ and state $s$. That probability is then divided by the *a priori* probability of the phone class $l$ at state $s$. A one-level HME performs the following computation:

$$P(l|x, s) = \sum_{i=1}^{C} P(l|c_i, x, s)P(c_i|x, s) \tag{1}$$

where $l = 1, ..., L$ indicates phoneme class, $c_i$ represents a local region in the input space, and $C$ is the number of regions. $P(c_i|x, s)$ can be viewed as a gating network, while $P(l|c_i, x, s)$ can be viewed as a local expert classifier (expert network) in the region $c_i$ [1]. In a two-level HME, each region $c_i$ is divided in turn into $C$ subregions. The term $P(l|c_i, x, s)$ is then computed in a similar manner to equation (1), and so on. If in some of these subregions there are no data available, we back off to the parent network.

## 3   TECHNICAL DETAILS

As in Jordan's paper, we use a generalized sigmoidal function to parameterize $P(c_i|x)$ as follows:

$$P(c_i|x) = \frac{e^{v_i^T x}}{\sum_j e^{v_j^T x}} \tag{2}$$

where $x$ can be the direct input (in a one-layer neural net) or the hidden layer vector (in a two-layer neural net), and $v_i, i = 1, ..., C$ are weights which need to train. Similarly, the local phoneme classifier in region $c_i$, $P(l|c_i, x)$, can be parameterized with a generalized

sigmoidal function also:

$$P(l|c_i, x) = \frac{e^{\theta_{li}^T x}}{\sum_j e^{\theta_{ji}^T x}} \qquad (3)$$

where $\theta_{ji}, j = 1, ..., L$ are weights. The whole system consists of two set of parameters: $v_i, i = 1, ..., C$ and $\theta_{ji}, j = 1, ..., L, \Theta = \{\theta_{ji}, v_i\}$. All parameters are estimated by using the EM algorithm.

The EM is an iterative approach to maximum likelihood estimation. Each iteration of an EM algorithm is composed of two steps: an Expectation (E) step and a Maximization (M) step. The M step involves the maximization of a likelihood function that is redefined in each iteration by the E step. Using the parameterizations in (2) and (3), we obtain the following iterative procedure for computing parameters $\Theta = \{v_i, \theta_{ji}\}$:

1. initialize $v_i^{(0)}$ and $\theta_{ji}^{(0)}$ for $i = 1, ..., C, j = 1, ..., L$.
2. E-step: In each iteration $n$, for each data pair $(x(t), l(t)), t = 1, ..., N$, compute

$$
\begin{aligned}
z_i(t)^{(n)} &= P(c_i|x(t), l(t), \Theta^{(n)}) \\
&= \frac{P(c_i|x(t), v_i^{(n)})P(l(t)|c_i, x(t), \theta_{l(t),i}^{(n)})}{\sum_k P(c_k|x(t), v_k^{(n)})P(l(t)|c_k, x(t), \theta_{l(t),k}^{(n)})}
\end{aligned}
\qquad (4)
$$

where $i = 1, ..., C$. $z_i(t)^{(n)}$ represents the probability of the data $t$ lying in the region $i$, given the current parameter estimation $\Theta^{(n)}$. It will be used as a weight for this data in the region $i$ in the M-step. The idea of "soft" splitting reflects that these weights are probabilities between 0 and 1, instead of a "hard"decision 0 or 1.
3. M-step:

$$\theta_i^{(n+1)} = \max_{\theta_i} \sum_t z_i(t)^{(n)} [\log \frac{e^{\theta_{l(t),i}^T x(t)}}{\sum_j e^{\theta_{ji}^T x(t)}}] \qquad (5)$$

$$v_{i=1,...,C}^{(n+1)} = \max_{v_1,...,v_C} \sum_t \sum_k z_k(t)^{(n)} \log \frac{e^{v_i^T x(t)}}{\sum_j e^{v_j^T x(t)}} \qquad (6)$$

4. Iterate until $\theta_{ji}, v_i$ converge.

The first maximization means fitting a generalized sigmoidal model (3) using the labeled data $(x(t), l(t))$ and weighting $z_i(t)^{(n)}$. The second one means fitting a generalized sigmoidal model (2) using inputs $x(t)$ and outputs $z_i(t)^{(n)}$. The criterion for fitting is the cross-entropy. Typically, the fitting can be solved by the Newton-Raphson method. However, it is quite expensive. Viewing this type of fitting as a multi-class classification task, we developed a technique to invert a generalized sigmoidal function more efficiently, which will be described in the following.

A common method in a multi-class classification is to divide the problem into many 2-class classifications. However, this method results in a positive and negative training unbalance usually. To avoid the positive and negative training unbalance, the following technique can be used to solve multi-class posterior probabilities simultaneously.

Suppose we have a labeled data set, $(x(t), l(t)), t = 1, ..., N$, where $l(t) \in \{1, ..., L\}$ is the label for t-th data. We use a generalized sigmoidal function to model the posterior

probability $P(l|x)$, where $l = 1, ..., L$ as follows:

$$P_l(x) = P(l|x) = \frac{e^{\theta_l^T x}}{\sum_k e^{\theta_k^T x}} \tag{7}$$

Obviously, since these probabilities sum up to one, we have

$$P_L(x) = 1 - \sum_{l=1}^{L-1} P_l(x). \tag{8}$$

Now, a training sample $x(t)$ with a class label $l(t)$ can be interpreted as:

$$P_l(x(t)) = \begin{cases} 0.9 & l = l(t) \\ \frac{0.1}{L-1} & l \neq l(t) \end{cases} \tag{9}$$

If we define

$$\theta_l^T x = \log \frac{P_l(x)}{P_L(x)} \tag{10}$$

equation (10) implies that

$$P_l(x) = \frac{e^{\theta_l^T x}}{\sum_l^L e^{\theta_l^T x}} \tag{11}$$

for $l = 1, ..., L$ with $\theta_L^T x = 0$. This expression is the generalized sigmoidal function in (7). This means, we can train parameters in (7) to satisfy Equation (10) from the data. Using a least squares criterion, the objective is

$$\min \sum_t \left[ \theta_l^T x(t) - \log \frac{P_l(x(t))}{P_L(x(t))} \right]^2 \tag{12}$$

for $l = 1, ..., L - 1$. Denote a data matrix as

$$X = \begin{bmatrix} x(1) \\ x(2) \\ \cdot \\ \cdot \\ \cdot \\ x(N) \end{bmatrix}$$

A least squares solution to (12) is

$$\theta_l = (\log a)(X^T X)^{-1} \left[ \sum_{l(t)=l} x(t) - \sum_{l(t)=L} x(t) \right] \tag{13}$$

for $l = 1, ..., L$, where $a = 9(L - 1)$. Substituting (13) into (11), we get

$$P_l(x) = \frac{a^{x^T (X^T X)^{-1} \sum_{l(t)=l} x(t)}}{\sum_k a^{x^T (X^T X)^{-1} \sum_{l(t)=k} x(t)}} \tag{14}$$

Equation (13) and (14) are very easy to compute. Basically, we only have to accumulate the matrix $X^T X$ and sum $x(t)$ into different classes $l = 1, ..., L$. We can obtain probabilities $P_l(x)$ by a single inversion of matrix $X^T X$ after a pass through the training data.

## 4  Relation to Other Work

The work reported here is very different from our previous work utilizing neural nets for CSR. There, a single segmental neural network (SNN) is used to model a complete phonetic segment [3]. Here, each HME estimates the probability density for each state of a phonetic HMM. The work here is more similar to that by Cohen *et al.* [4], the major difference being that in [4], a single very large neural net is used to perform the probability density modeling. The training of such a large network requires the use of a specialized parallel processing machine, so that the training can be done in a reasonable amount of time. By using the HME method and dividing the problem into many smaller problems, we are able to perform the needed training computation on regular workstations.

Most of the previous work on CD-HMM work has utilized mixtures of gaussians to estimate the probability densities of an HMM. Since a multilayer feedforward neural network is a universal continuous function approximator, we decided to explore the use of neural nets as an alternative approach for continuous density estimation.

## 5  Experimental Results

|  | Word Error Rate |
| --- | --- |
| HMM | 7.8 |
| SNN | 8.5 |
| HMM+SNN | 7.1 |
| HME | 7.6 |
| HME + HMM | 6.8 |
| Prior-modified HME + HMM | 6.2 |

Table 1: Error Rates for the ARPA WSJ 5K Development Test, Trigram Grammar

|  | Word Error Rate |
| --- | --- |
| HMM | 9.5 |
| HME + HMM | 8.7 |

Table 2: Error Rates for the ARPA WSJ 40K Test Set, Trigram Grammar

In our initial application of the HME method to large-vocabulary CSR, we used phonetic context-independent HMEs to estimate the likelihoods at each state of 5-state HMMs. We implemented a two-level HME, with the input space divided into 46 regions, and each of those regions is further divided into 46 subregions. The initial divisions were accomplished by supervised training, with each division trained to one of the 46 phonemes in the system. All gating and local expert networks in the HME had identical structures — a two-layer generalized sigmoidal network. The whole HME system was implemented within an N-best paradigm [3], where the recognized sequence was obtained as a result of a rescoring of an N-best list obtained from our baseline BYBLOS system (tied-mixture HMM) with a statistical trigram grammar.

We then built a context-dependent HME system based on the structure of the context-independent HME models described above. For each state, the whole training data was divided into 46 parts according to its left or right context. Then for each context, a separate HME model was built for that context. To be computationally feasible, we used only one-level HMEs here. We first experimented using a left-context and right-context model.

We tested the HME implementation on the ARPA 5,000-word Wall Street Journal corpus (WSJ1, H2 dev set). We report the word error rates on the same test set for a number of different systems. Table 1 shows the word error rates for i) the baseline HMM system; ii) the segment-based neural net system (SNN) iii) the hybrid SNN/HMM system iv) a HME system alone. v) a HME system combined with HMM; vi) a HME +HMM system with modified priors.

From Table 1, The performance of the baseline tied-mixture HMM is 7.8%. The performance of the SNN system (8.5%) is comparable to the HMM alone. We see that the performance of a HME (7.6%) is as good as the HMM system, which is better than the SNN system. When combined with the baseline HMM system, the HME and SNN both improve performance over the HMM alone about 10% from 7.8% to 6.8% and from 7.8% to 7.1% respectively. We found out that the improvement could be made larger for a hybrid HME/HMM by adjusting the context-dependent priors with the context-independent priors, and then smooth context-dependent models with a context-independent model.

More specifically, in a context-dependent HME model, we usually estimate the posterior probability phoneme $l$, $P(l|c, x, s)$, given left or right context $c$ and the acoustic input $x$ in a particular state $s$. Because the samples may be sparse for many of context models, it is necessary to regularize (smooth) context-dependent models with a context-independent model, where there is much more data available. However, since the two models have different priors: $P(l|c, s)$ in a context-dependent model and $P(l|s)$ in a context-independent model, a simple interpolation between the two models which is $P(l|c, x, s) = \frac{P(x|l,c,s)P(l|c,s)}{P(x|c,s)}$ in a context-dependent model and $P(l|x, s) = \frac{P(x|l,s)P(l|s)}{P(x|s)}$ in a context-independent model is inconsistent. To scale the context-dependent priors $P(l|c, s)$ with a context-independent prior $P(l|s)$, we weighted each input data point $x$ with the weight $\frac{P(l|s)}{P(l|c,s)}$ for a prior adjusting. After this modification, a context-dependent HME actually estimates $\frac{P(x|l,c,s)P(l|s)}{P(x|s)}$. It combines better with a context-independent model. For the same experiment we showed in Table 1, the word error for the HME (with HMM) droped from 6.8% to 6.2% when priors were modified. For this 5,000-word development set, we got a total of about 20% word error reduction over the tied-mixture HMM system using a HME-based neural network system.

We then switched our experiment domain from a 5,000-word to 40,000-word the test set. During this year, the BYBLOS system has been improved from a tied-mixture system to a continuous density system. We also switched to using this new continuous density BYBLOS in our hybrid HME/HMM system. The language model used here was a 40,000-word trigram grammar. The result is shown in Table 2.

From Table 2, we see that there is about a 10% word error rate reduction over the continuous density HMM system by combining a context-dependent HME system. Compared with the 20% improvement over the tied-mixture system we made for the 5,000-word development set, the improvement over the continuous density system in this 40,000-word

development is less. This may be due to the big improvement of the HMM system itself.

## 6  CONCLUSIONS

The method of hierarchical mixtures of experts can be used as a continous density estimator to speech recognition. Experimental results showed that estimations from this approach are consistent with the estimations from the HMM system. The frame-based neural net system using hierarchical mixtures of experts improves the performance of both the state-of-the-art tied mixture HMM system and the continuous density HMM system. The HME system itself has the same performance as the state-of-the-art tied mixture HME system.

## 7  Acknowledgments

This work was funded by the Advanced Research Projects Agency of the Department of Defense.

## References

[1] Michael Jordan, "Hierarchical Mixtures of Experts and the EM Algorithm," *Neural Computation*, 1994, in press.

[2] D. Pallett, J. Fiscus, W. Fisher, J. Garofolo, B. Lund, and M. Pryzbocki, "1993 Benchmark Tests for the ARPA Spoken Language Program," *Proc. ARPA Human Language Technology Workshop*, Plainsboro, NJ, Morgan Kaufman Publishers, 1994.

[3] G. Zavaliagkos, Y. Zhao, R. Schwartz and J. Makhoul, "A Hybrid Neural Net System for State-of-the-Art Continuous Speech Recognition," in *Advances in Neural Information Processing Systems 5*, S. J. Hanson, J. D. Cowan and C. L. Giles, eds., Morgan Kaufmann Publishers, 1993.

[4] M. Cohen, H. Franco, N. Morgan, D. Rumelhart and V. Abrash, "Context-Dependent Multiple Distribution Phonetic Modeling with MLPS," in *Advances in Neural Information Processing Systems 5*, S. J. Hanson, J. D. Cowan and C. L. Giles, eds., Morgan Kaufmann Publishers, 1993.

